# Algorithms for Hyper-Parameter Optimization

**James Bergstra**
The Rowland Institute
Harvard University
bergstra@rowland.harvard.edu

**Rémi Bardenet**
Laboratoire de Recherche en Informatique
Université Paris-Sud
bardenet@lri.fr

**Yoshua Bengio**
Dépt. d'Informatique et Recherche Opérationelle
Université de Montréal
yoshua.bengio@umontreal.ca

**Balázs Kégl**
Linear Accelerator Laboratory
Université Paris-Sud, CNRS
balazs.kegl@gmail.com

## Abstract

Several recent advances to the state of the art in image classification benchmarks have come from better configurations of existing techniques rather than novel approaches to feature learning. Traditionally, hyper-parameter optimization has been the job of humans because they can be very efficient in regimes where only a few trials are possible. Presently, computer clusters and GPU processors make it possible to run more trials and we show that algorithmic approaches can find better results. We present hyper-parameter optimization results on tasks of training neural networks and deep belief networks (DBNs). We optimize hyper-parameters using random search and two new greedy sequential methods based on the expected improvement criterion. Random search has been shown to be sufficiently efficient for learning neural networks for several datasets, but we show it is unreliable for training DBNs. The sequential algorithms are applied to the most difficult DBN learning problems from [1] and find significantly better results than the best previously reported. This work contributes novel techniques for making response surface models $P(y|x)$ in which many elements of hyper-parameter assignment $(x)$ are known to be irrelevant given particular values of other elements.

## 1 Introduction

Models such as Deep Belief Networks (DBNs) [2], stacked denoising autoencoders [3], convolutional networks [4], as well as classifiers based on sophisticated feature extraction techniques have from ten to perhaps fifty hyper-parameters, depending on how the experimenter chooses to parametrize the model, and how many hyper-parameters the experimenter chooses to fix at a reasonable default. The difficulty of tuning these models makes published results difficult to reproduce and extend, and makes even the original investigation of such methods more of an art than a science.

Recent results such as [5], [6], and [7] demonstrate that the challenge of hyper-parameter optimization in large and multilayer models is a direct impediment to scientific progress. These works have advanced state of the art performance on image classification problems by more concerted hyper-parameter optimization in simple algorithms, rather than by innovative modeling or machine learning strategies. It would be wrong to conclude from a result such as [5] that feature learning is useless. Instead, hyper-parameter optimization should be regarded as a formal outer loop in the learning process. A learning algorithm, as *a functional from data to classifier* (taking classification problems as an example), includes a budgeting choice of how many CPU cycles are to be spent on hyper-parameter exploration, and how many CPU cycles are to be spent evaluating each hyper-parameter choice (i.e. by tuning the regular parameters). The results of [5] and [7] suggest that with current generation hardware such as large computer clusters and GPUs, the optimal alloca-

tion of CPU cycles includes more hyper-parameter exploration than has been typical in the machine learning literature.

Hyper-parameter optimization is the problem of optimizing a loss function over a graph-structured configuration space. In this work we restrict ourselves to tree-structured configuration spaces. Configuration spaces are tree-structured in the sense that some leaf variables (e.g. the number of hidden units in the 2nd layer of a DBN) are only well-defined when node variables (e.g. a discrete choice of how many layers to use) take particular values. Not only must a hyper-parameter optimization algorithm optimize over variables which are discrete, ordinal, and continuous, but it must simultaneously choose which variables to optimize.

In this work we define a configuration space by a generative process for drawing valid samples. Random search is the algorithm of drawing hyper-parameter assignments from that process and evaluating them. Optimization algorithms work by identifying hyper-parameter assignments that could have been drawn, and that appear promising on the basis of the loss function's value at other points. This paper makes two contributions: 1) Random search is competitive with the manual optimization of DBNs in [1], and 2) Automatic sequential optimization outperforms both manual and random search.

Section 2 covers sequential model-based optimization, and the expected improvement criterion. Section 3 introduces a Gaussian Process based hyper-parameter optimization algorithm. Section 4 introduces a second approach based on adaptive Parzen windows. Section 5 describes the problem of DBN hyper-parameter optimization, and shows the efficiency of random search. Section 6 shows the efficiency of sequential optimization on the two hardest datasets according to random search. The paper concludes with discussion of results and concluding remarks in Section 7 and Section 8.

## 2 Sequential Model-based Global Optimization

Sequential Model-Based Global Optimization (SMBO) algorithms have been used in many applications where evaluation of the fitness function is expensive [8, 9]. In an application where the true fitness function $f : \mathcal{X} \to \mathbb{R}$ is costly to evaluate, model-based algorithms approximate $f$ with a surrogate that is cheaper to evaluate. Typically the inner loop in an SMBO algorithm is the numerical optimization of this surrogate, or some transformation of the surrogate. The point $x^*$ that maximizes the surrogate (or its transformation) becomes the proposal for where the true function $f$ should be evaluated. This active-learning-like algorithm template is summarized in Figure 1. SMBO algorithms differ in what criterion they optimize to obtain $x^*$ given a model (or surrogate) of $f$, and in they model $f$ via observation history $\mathcal{H}$.

$\text{SMBO}(f, M_0, T, S)$

| | |
|---|---|
| 1 | $\mathcal{H} \leftarrow \emptyset,$ |
| 2 | For $t \leftarrow 1$ **to** $T,$ |
| 3 | $x^* \leftarrow \text{argmin}_x \; S(x, M_{t-1}),$ |
| 4 | Evaluate $f(x^*),$ ▷ *Expensive step* |
| 5 | $\mathcal{H} \leftarrow \mathcal{H} \cup (x^*, f(x^*)),$ |
| 6 | Fit a new model $M_t$ to $\mathcal{H}.$ |
| 7 | **return** $\mathcal{H}$ |

Figure 1: The pseudo-code of generic Sequential Model-Based Optimization.

The algorithms in this work optimize the criterion of Expected Improvement (EI) [10]. Other criteria have been suggested, such as Probability of Improvement and Expected Improvement [10], minimizing the Conditional Entropy of the Minimizer [11], and the bandit-based criterion described in [12]. We chose to use the EI criterion in our work because it is intuitive, and has been shown to work well in a variety of settings. We leave the systematic exploration of improvement criteria for future work. Expected improvement is the expectation under some model $M$ of $f : \mathcal{X} \to \mathbb{R}^N$ that $f(x)$ will exceed (negatively) some threshold $y^*$:

$$\text{EI}_{y^*}(x) := \int_{-\infty}^{\infty} \max(y^* - y, 0) p_M(y|x) dy. \tag{1}$$

The contribution of this work is two novel strategies for approximating $f$ by modeling $\mathcal{H}$: a hierarchical Gaussian Process and a tree-structured Parzen estimator. These are described in Section 3 and Section 4 respectively.

# 3 The Gaussian Process Approach (GP)

Gaussian Processes have long been recognized as a good method for modeling loss functions in model-based optimization literature [13]. Gaussian Processes (GPs, [14]) are priors over functions that are *closed under sampling*, which means that if the prior distribution of $f$ is believed to be a GP with mean 0 and kernel $k$, the conditional distribution of $f$ knowing a sample $\mathcal{H} = (x_i, f(x_i))_{i=1}^n$ of its values is also a GP, whose mean and covariance function are analytically derivable. GPs with generic mean functions can in principle be used, but it is simpler and sufficient for our purposes to only consider zero mean processes. We do this by centering the function values in the considered data sets. Modelling e.g. linear trends in the GP mean leads to undesirable extrapolation in unexplored regions during SMBO [15].

The above mentioned closedness property, along with the fact that GPs provide an assessment of prediction uncertainty incorporating the effect of data scarcity, make the GP an elegant candidate for both finding candidate $x^*$ (Figure 1, step 3) and fitting a model $M_t$ (Figure 1, step 6). The runtime of each iteration of the GP approach scales cubically in $|\mathcal{H}|$ and linearly in the number of variables being optimized, however the expense of the function evaluations $f(x^*)$ typically dominate even this cubic cost.

## 3.1 Optimizing EI in the GP

We model $f$ with a GP and set $y^*$ to the best value found after observing $\mathcal{H}$: $y^* = \min\{f(x_i), 1 \leq i \leq n\}$. The model $p_M$ in (1) is then the posterior GP knowing $\mathcal{H}$. The EI function in (1) encapsulates a compromise between regions where the mean function is close to or better than $y^*$ and under-explored regions where the uncertainty is high.

EI functions are usually optimized with an exhaustive grid search over the input space, or a Latin Hypercube search in higher dimensions. However, some information on the landscape of the EI criterion can be derived from simple computations [16]: 1) it is always non-negative and zero at training points from $\mathcal{D}$, 2) it inherits the smoothness of the kernel $k$, which is in practice often at least once differentiable, and noticeably, 3) the EI criterion is likely to be highly multi-modal, especially as the number of training points increases. The authors of [16] used the preceding remarks on the landscape of EI to design an evolutionary algorithm with mixture search, specifically aimed at optimizing EI, that is shown to outperform exhaustive search for a given budget in EI evaluations. We borrow here their approach and go one step further. We keep the Estimation of Distribution (EDA, [17]) approach on the discrete part of our input space (categorical and discrete hyper-parameters), where we sample candidate points according to binomial distributions, while we use the Covariance Matrix Adaptation - Evolution Strategy (CMA-ES, [18]) for the remaining part of our input space (continuous hyper-parameters). CMA-ES is a state-of-the-art gradient-free evolutionary algorithm for optimization on continuous domains, which has been shown to outperform the Gaussian search EDA. Notice that such a gradient-free approach allows non-differentiable kernels for the GP regression. We do not take on the use of mixtures in [16], but rather restart the local searches several times, starting from promising places. The use of tesselations suggested by [16] is prohibitive here, as our task often means working in more than 10 dimensions, thus we start each local search at the center of mass of a simplex with vertices randomly picked among the training points.

Finally, we remark that all hyper-parameters are not relevant for each point. For example, a DBN with only one hidden layer does not have parameters associated to a second or third layer. Thus it is not enough to place one GP over the entire space of hyper-parameters. We chose to group the hyper-parameters by common use in a tree-like fashion and place different independent GPs over each group. As an example, for DBNs, this means placing one GP over common hyper-parameters, including categorical parameters that indicate what are the *conditional* groups to consider, three GPs on the parameters corresponding to each of the three layers, and a few 1-dimensional GPs over individual conditional hyper-parameters, like ZCA energy (see Table 1 for DBN parameters).

# 4   Tree-structured Parzen Estimator Approach (TPE)

Anticipating that our hyper-parameter optimization tasks will mean high dimensions and small fitness evaluation budgets, we now turn to another modeling strategy and EI optimization scheme for the SMBO algorithm. Whereas the Gaussian-process based approach modeled $p(y|x)$ directly, this strategy models $p(x|y)$ and $p(y)$.

Recall from the introduction that the configuration space $\mathcal{X}$ is described by a graph-structured generative process (e.g. first choose a number of DBN layers, then choose the parameters for each). The tree-structured Parzen estimator (TPE) models $p(x|y)$ by transforming that generative process, replacing the distributions of the configuration prior with non-parametric densities. In the experimental section, we will see that the configuation space is described using uniform, log-uniform, quantized log-uniform, and categorical variables. In these cases, the TPE algorithm makes the following replacements: uniform $\rightarrow$ truncated Gaussian mixture, log-uniform $\rightarrow$ exponentiated truncated Gaussian mixture, categorical $\rightarrow$ re-weighted categorical. Using different observations $\{x^{(1)}, ..., x^{(k)}\}$ in the non-parametric densities, these substitutions represent a *learning algorithm* that can produce a variety of densities over the configuration space $\mathcal{X}$. The TPE defines $p(x|y)$ using two such densities:

$$p(x|y) = \begin{cases} \ell(x) & \text{if } y < y^* \\ g(x) & \text{if } y \geq y^*, \end{cases} \tag{2}$$

where $\ell(x)$ is the density formed by using the observations $\{x^{(i)}\}$ such that corresponding loss $f(x^{(i)})$ was less than $y^*$ and $g(x)$ is the density formed by using the remaining observations. Whereas the GP-based approach favoured quite an aggressive $y^*$ (typically less than the best observed loss), the TPE algorithm depends on a $y^*$ that is larger than the best observed $f(x)$ so that some points can be used to form $\ell(x)$. The TPE algorithm chooses $y^*$ to be some quantile $\gamma$ of the observed $y$ values, so that $p(y < y^*) = \gamma$, but no specific model for $p(y)$ is necessary. By maintaining sorted lists of observed variables in $\mathcal{H}$, the runtime of each iteration of the TPE algorithm can scale linearly in $|\mathcal{H}|$ and linearly in the number of variables (dimensions) being optimized.

## 4.1   Optimizing EI in the TPE algorithm

The parametrization of $p(x, y)$ as $p(y)p(x|y)$ in the TPE algorithm was chosen to facilitate the optimization of EI.

$$\text{EI}_{y^*}(x) = \int_{-\infty}^{y^*} (y^* - y)p(y|x)dy = \int_{-\infty}^{y^*} (y^* - y)\frac{p(x|y)p(y)}{p(x)}dy \tag{3}$$

By construction, $\gamma = p(y < y^*)$ and $p(x) = \int_{\mathbb{R}} p(x|y)p(y)dy = \gamma\ell(x) + (1 - \gamma)g(x)$. Therefore

$$\int_{-\infty}^{y^*} (y^* - y)p(x|y)p(y)dy = \ell(x) \int_{-\infty}^{y^*} (y^* - y)p(y)dy = \gamma y^* \ell(x) - \ell(x) \int_{-\infty}^{y^*} p(y)dy,$$

so that finally $EI_{y^*}(x) = \frac{\gamma y^* \ell(x) - \ell(x) \int_{-\infty}^{y^*} p(y)dy}{\gamma\ell(x) + (1 - \gamma)g(x)} \propto \left(\gamma + \frac{g(x)}{\ell(x)}(1 - \gamma)\right)^{-1}$. This last expression shows that to maximize improvement we would like points $x$ with high probability under $\ell(x)$ and low probability under $g(x)$. The tree-structured form of $\ell$ and $g$ makes it easy to draw many candidates according to $\ell$ and evaluate them according to $g(x)/\ell(x)$. On each iteration, the algorithm returns the candidate $x^*$ with the greatest EI.

## 4.2   Details of the Parzen Estimator

The models $\ell(x)$ and $g(x)$ are hierarchical processes involving discrete-valued and continuous-valued variables. The Adaptive Parzen Estimator yields a model over $\mathcal{X}$ by placing density in the vicinity of $K$ observations $\mathcal{B} = \{x^{(1)}, ..., x^{(K)}\} \subset \mathcal{H}$. Each continuous hyper-parameter was specified by a uniform prior over some interval $(a, b)$, or a Gaussian, or a log-uniform distribution. The TPE substitutes an equally-weighted mixture of that prior with Gaussians centered at each of the $x^{(i)} \in \mathcal{B}$. The standard deviation of each Gaussian was set to the greater of the distances to the left and right neighbor, but clipped to remain in a reasonable range. In the case of the uniform, the points $a$ and $b$ were considered to be potential neighbors. For discrete variables, supposing the prior was a vector of $N$ probabilities $p_i$, the posterior vector elements were proportional to $Np_i + C_i$ where $C_i$ counts the occurrences of choice $i$ in $\mathcal{B}$. The log-uniform hyper-parameters were treated as uniforms in the log domain.

Table 1: Distribution over DBN hyper-parameters for random sampling. Options separated by "or" such as pre-processing (and including the random seed) are weighted equally. Symbol $U$ means uniform, $\mathcal{N}$ means Gaussian-distributed, and $\log U$ means uniformly distributed in the log-domain. CD (also known as CD-1) stands for contrastive divergence, the algorithm used to initialize the layer parameters of the DBN.

| Whole model | | Per-layer | |
|---|---|---|---|
| Parameter | Prior | Parameter | Prior |
| pre-processing | raw or ZCA | n. hidden units | $\log U(128, 4096)$ |
| ZCA energy | $U(.5, 1)$ | $W$ init | $U(-a, a)$ or $\mathcal{N}(0, a^2)$ |
| random seed | 5 choices | $a$ | algo A or B (see text) |
| classifier learn rate | $\log U(0.001, 10)$ | algo A coef | $U(.2, 2)$ |
| classifier anneal start | $\log U(100, 10^4)$ | CD epochs | $\log U(1, 10^4)$ |
| classifier $\ell_2$-penalty | 0 or $\log U(10^{-7}, 10^{-4})$ | CD learn rate | $\log U(10^{-4}, 1)$ |
| n. layers | 1 to 3 | CD anneal start | $\log U(10, 10^4)$ |
| batch size | 20 or 100 | CD sample data | yes or no |

## 5 Random Search for Hyper-Parameter Optimization in DBNs

One simple, but recent step toward formalizing hyper-parameter optimization is the use of random search [5]. [19] showed that random search was much more efficient than grid search for optimizing the parameters of one-layer neural network classifiers. In this section, we evaluate random search for DBN optimization, compared with the sequential grid-assisted manual search carried out in [1].

We chose the prior listed in Table 1 to define the search space over DBN configurations. The details of the datasets, the DBN model, and the greedy layer-wise training procedure based on CD are provided in [1]. This prior corresponds to the search space of [1] except for the following differences: (a) we allowed for ZCA pre-processing [20], (b) we allowed for each layer to have a different size, (c) we allowed for each layer to have its own training parameters for CD, (d) we allowed for the possibility of treating the continuous-valued data as either as Bernoulli means (more theoretically correct) or Bernoulli samples (more typical) in the CD algorithm, and (e) we did not discretize the possible values of real-valued hyper-parameters. These changes expand the hyper-parameter search problem, while maintaining the original hyper-parameter search space as a subset of the expanded search space.

The results of this preliminary random search are in Figure 2. Perhaps surprisingly, the result of manual search can be reliably matched with 32 random trials for several datasets. The efficiency of random search in this setting is explored further in [21]. Where random search results match human performance, it is not clear from Figure 2 whether the reason is that it searched the original space as efficiently, or that it searched a larger space where good performance is easier to find. But the objection that random search is somehow cheating by searching a larger space is backward – the search space outlined in Table 1 is a natural description of the hyper-parameter optimization problem, and the restrictions to that space by [1] were presumably made to simplify the search problem and make it tractable for grid-search assisted manual search. Critically, both methods train DBNs on the same datasets.

The results in Figure 2 indicate that hyper-parameter optimization is harder for some datasets. For example, in the case of the "MNIST rotated background images" dataset (**MRBI**), random sampling appears to converge to a maximum relatively quickly (best models among experiments of 32 trials show little variance in performance), but this plateau is lower than what was found by manual search. In another dataset (**convex**), the random sampling procedure exceeds the performance of manual search, but is slow to converge to any sort of plateau. There is considerable variance in generalization when the best of 32 models is selected. This slow convergence indicates that better performance is probably available, but we need to search the configuration space more efficiently to find it. The remainder of this paper explores sequential optimization strategies for hyper-parameter optimization for these two datasets: **convex** and **MRBI**.

## 6 Sequential Search for Hyper-Parameter Optimization in DBNs

We validated our GP approach of Section 3.1 by comparing with random sampling on the Boston Housing dataset, a regression task with 506 points made of 13 scaled input variables and a scalar

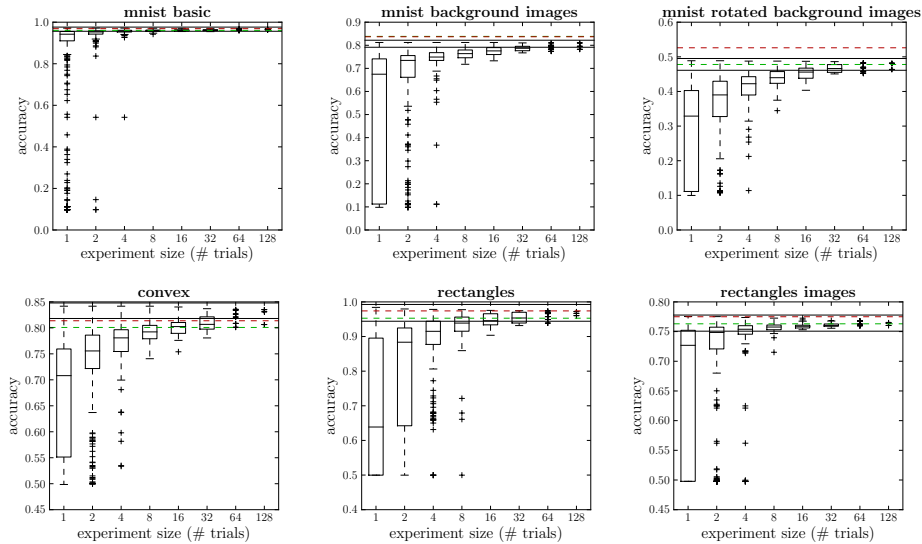

Figure 2: Deep Belief Network (DBN) performance according to random search. Random search is used to explore up to 32 hyper-parameters (see Table 1). Results found using a grid-search-assisted manual search over a similar domain with an average 41 trials are given in green (1-layer DBN) and red (3-layer DBN). Each box-plot (for $N = 1, 2, 4, ...$) shows the distribution of test set performance when the best model among $N$ random trials is selected. The datasets "convex" and "mnist rotated background images" are used for more thorough hyper-parameter optimization.

regressed output. We trained a Multi-Layer Perceptron (MLP) with 10 hyper-parameters, including learning rate, $\ell_1$ and $\ell_2$ penalties, size of hidden layer, number of iterations, whether a PCA pre-processing was to be applied, whose energy was the only conditional hyper-parameter [22]. Our results are depicted in Figure 3. The first 30 iterations were made using random sampling, while from the 30th on, we differentiated the random samples from the GP approach trained on the updated history. The experiment was repeated 20 times. Although the number of points is particularly small compared to the dimensionality, the surrogate modelling approach finds noticeably better points than random, which supports the application of SMBO approaches to more ambitious tasks and datasets.

Applying the GP to the problem of optimizing DBN performance, we allowed 3 random restarts to the CMA+ES algorithm per proposal $x^*$, and up to 500 iterations of conjugate gradient method in fitting the length scales of the GP. The squared exponential kernel [14] was used for every node. The CMA-ES part of GPs dealt with boundaries using a penalty method, the binomial sampling part dealt with it by nature. The GP algorithm was initialized with 30 randomly sampled points in $\mathcal{H}$. After 200 trials, the prediction of a point $x^*$ using this GP took around 150 seconds.

For the TPE-based algorithm, we chose $\gamma = 0.15$ and picked the best among 100 candidates drawn from $\ell(x)$ on each iteration as the proposal $x^*$. After 200 trials, the prediction of a point $x^*$ using this TPE algorithm took around 10 seconds. TPE was allowed to grow past the initial bounds used with for random sampling in the course of optimization, whereas the GP and random search were restricted to stay within the initial bounds throughout the course of optimization. The TPE algorithm was also initialized with the same 30 randomly sampled points as were used to seed the GP.

## 6.1 Parallelizing Sequential Search

Both the GP and TPE approaches were actually run asynchronously in order to make use of multiple compute nodes and to avoid wasting time waiting for trial evaluations to complete. For the GP approach, the so-called *constant liar* approach was used: each time a candidate point $x^*$ was proposed, a fake fitness evaluation equal to the mean of the $y$'s within the training set $\mathcal{D}$ was assigned temporarily, until the evaluation completed and reported the actual loss $f(x^*)$. For the TPE approach, we simply ignored recently proposed points and relied on the stochasticity of draws from $\ell(x)$ to provide different candidates from one iteration to the next. The consequence of parallelization is that each proposal $x^*$ is based on less feedback. This makes search less efficient, though faster in terms of wall time.

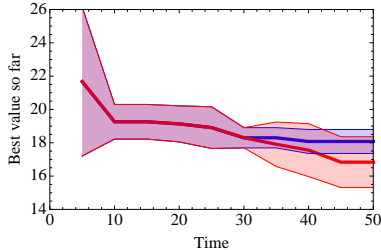

Figure 3: After time 30, GP optimizing the MLP hyper-parameters on the Boston Housing regression task. Best minimum found so far every 5 iterations, against time. Red = GP, Blue = Random. Shaded areas = one-sigma error bars.

| | convex | MRBI |
|---|---|---|
| TPE | **14.13** $\pm 0.30$ % | **44.55** $\pm 0.44$% |
| GP | $16.70 \pm 0.32$% | $47.08 \pm 0.44$% |
| Manual | $18.63 \pm 0.34$% | $47.39 \pm 0.44$% |
| Random | $18.97 \pm 0.34$ % | $50.52 \pm 0.44$% |

Table 2: The test set classification error of the best model found by each search algorithm on each problem. Each search algorithm was allowed up to 200 trials. The manual searches used 82 trials for **convex** and 27 trials **MRBI**.

Runtime per trial was limited to 1 hour of GPU computation regardless of whether execution was on a GTX 285, 470, 480, or 580. The difference in speed between the slowest and fastest machine was roughly two-fold in theory, but the actual efficiency of computation depended also on the load of the machine and the configuration of the problem (the relative speed of the different cards is different in different hyper-parameter configurations). With the parallel evaluation of up to five proposals from the GP and TPE algorithms, each experiment took about 24 hours of wall time using five GPUs.

## 7 Discussion

The trajectories ($\mathcal{H}$) constructed by each algorithm up to 200 steps are illustrated in Figure 4, and compared with random search and the manual search carried out in [1]. The generalization scores of the best models found using these algorithms and others are listed in Table 2. On the **convex** dataset (2-way classification), both algorithms converged to a validation score of 13% error. In generalization, TPE's best model had 14.1% error and GP's best had 16.7%. TPE's best was significantly better than both manual search (19%) and random search with 200 trials (17%). On the **MRBI** dataset (10-way classification), random search was the worst performer (50% error), the GP approach and manual search approximately tied (47% error), while the TPE algorithm found a new best result (44% error). The models found by the TPE algorithm in particular are better than previously found ones on both datasets. The GP and TPE algorithms were slightly less efficient than manual search: GP and EI identified performance on par with manual search within 80 trials, the manual search of [1] used 82 trials for **convex** and 27 trials for **MRBI**.

There are several possible reasons for why the TPE approach outperformed the GP approach in these two datasets. Perhaps the inverse factorization of $p(x|y)$ is more accurate than the $p(y|x)$ in the Gaussian process. Perhaps, conversely, the exploration induced by the TPE's lack of accuracy turned out to be a good heuristic for search. Perhaps the hyper-parameters of the GP approach itself were not set to correctly trade off exploitation and exploration in the DBN configuration space. More empirical work is required to test these hypotheses. Critically though, all four SMBO runs matched or exceeded both random search and a careful human-guided search, which are currently the state of the art methods for hyper-parameter optimization.

The GP and TPE algorithms work well in both of these settings, but there are certainly settings in which these algorithms, and in fact SMBO algorithm in general, would not be expected to do well. Sequential optimization algorithms work by leveraging structure in observed $(x, y)$ pairs. It is possible for SMBO to be arbitrarily bad with a bad choice of $p(y|x)$. It is also possible to be slower than random sampling at finding a global optimum with a apparently good $p(y|x)$, if it extracts structure in $\mathcal{H}$ that leads only to a local optimum.

## 8 Conclusion

This paper has introduced two sequential hyper-parameter optimization algorithms, and shown them to meet or exceed human performance and the performance of a brute-force random search in two difficult hyper-parameter optimization tasks involving DBNs. We have relaxed standard constraints (e.g. equal layer sizes at all layers) on the search space, and fall back on a more natural hyper-parameter space of 32 variables (including both discrete and continuous variables) in which many

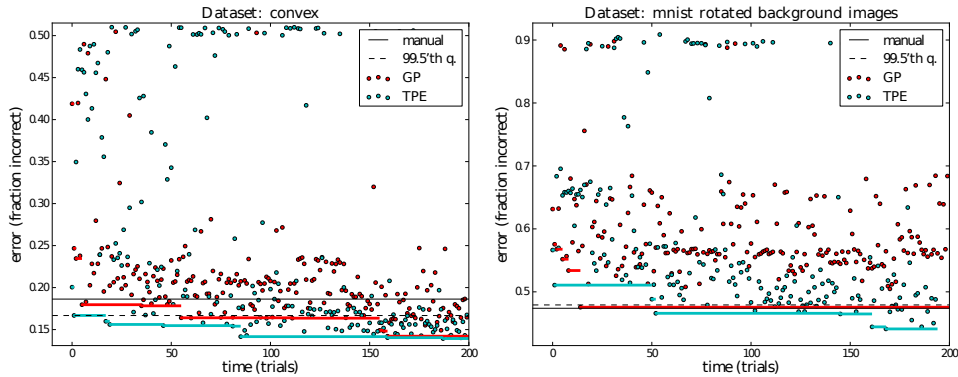

Figure 4: Efficiency of Gaussian Process-based (GP) and graphical model-based (TPE) sequential optimization algorithms on the task of optimizing the validation set performance of a DBN of up to three layers on the **convex** task (left) and the **MRBI** task (right). The dots are the elements of the trajectory $\mathcal{H}$ produced by each SMBO algorithm. The solid coloured lines are the validation set accuracy of the best trial found before each point in time. Both the TPE and GP algorithms make significant advances from their random initial conditions, and substantially outperform the manual and random search methods. A 95% confidence interval about the best validation means on the **convex** task extends 0.018 above and below each point, and on the **MRBI** task extends 0.021 above and below each point. The solid black line is the test set accuracy obtained by domain experts using a combination of grid search and manual search [1]. The dashed line is the 99.5% quantile of validation performance found among trials sampled from our prior distribution (see Table 1), estimated from 457 and 361 random trials on the two datasets respectively.

variables are sometimes irrelevant, depending on the value of other parameters (e.g. the number of layers). In this 32-dimensional search problem, the TPE algorithm presented here has uncovered new best results on both of these datasets that are significantly better than what DBNs were previously believed to achieve. Moreover, the GP and TPE algorithms are practical: the optimization for each dataset was done in just 24 hours using five GPU processors. Although our results are only for DBNs, our methods are quite general, and extend naturally to any hyper-parameter optimization problem in which the hyper-parameters are drawn from a measurable set.

We hope that our work may spur researchers in the machine learning community to treat the hyper-parameter optimization strategy as an interesting and important component of all learning algorithms. The question of "How well does a DBN do on the **convex** task?" is not a fully specified, empirically answerable question – different approaches to hyper-parameter optimization will give different answers. Algorithmic approaches to hyper-parameter optimization make machine learning results easier to disseminate, reproduce, and transfer to other domains. The specific algorithms we have presented here are also capable, at least in some cases, of finding better results than were previously known. Finally, powerful hyper-parameter optimization algorithms broaden the horizon of models that can realistically be studied; researchers need not restrict themselves to systems of a few variables that can readily be tuned by hand.

The TPE algorithm presented in this work, as well as parallel evaluation infrastructure, is available as BSD-licensed free open-source software, which has been designed not only to reproduce the results in this work, but also to facilitate the application of these and similar algorithms to other hyper-parameter optimization problems.[1]

### Acknowledgements

This work was supported by the National Science and Engineering Research Council of Canada, Compute Canada, and by the ANR-2010-COSI-002 grant of the French National Research Agency. GPU implementations of the DBN model were provided by Theano [23].

## Footnotes

[1]"Hyperopt" software package: `https://github.com/jaberg/hyperopt`

# References

[1] H. Larochelle, D. Erhan, A. Courville, J. Bergstra, and Y. Bengio. An empirical evaluation of deep architectures on problems with many factors of variation. In *ICML 2007*, pages 473–480, 2007.

[2] G. E. Hinton, S. Osindero, and Y. Teh. A fast learning algorithm for deep belief nets. *Neural Computation*, 18:1527–1554, 2006.

[3] P. Vincent, H. Larochelle, I. Lajoie, Y. Bengio, and P. A. Manzagol. Stacked denoising autoencoders: Learning useful representations in a deep network with a local denoising criterion. *Machine Learning Research*, 11:3371–3408, 2010.

[4] Y. LeCun, L. Bottou, Y. Bengio, and P. Haffner. Gradient-based learning applied to document recognition. *Proceedings of the IEEE*, 86(11):2278–2324, November 1998.

[5] Nicolas Pinto, David Doukhan, James J. DiCarlo, and David D. Cox. A high-throughput screening approach to discovering good forms of biologically inspired visual representation. *PLoS Comput Biol*, 5(11):e1000579, 11 2009.

[6] A. Coates, H. Lee, and A. Ng. An analysis of single-layer networks in unsupervised feature learning. NIPS Deep Learning and Unsupervised Feature Learning Workshop, 2010.

[7] A. Coates and A. Y. Ng. The importance of encoding versus training with sparse coding and vector quantization. In *Proceedings of the Twenty-eighth International Conference on Machine Learning (ICML-11)*, 2010.

[8] F. Hutter. *Automated Configuration of Algorithms for Solving Hard Computational Problems*. PhD thesis, University of British Columbia, 2009.

[9] F. Hutter, H. Hoos, and K. Leyton-Brown. Sequential model-based optimization for general algorithm configuration. In *LION-5*, 2011. Extended version as UBC Tech report TR-2010-10.

[10] D.R. Jones. A taxonomy of global optimization methods based on response surfaces. *Journal of Global Optimization*, 21:345–383, 2001.

[11] J. Villemonteix, E. Vazquez, and E. Walter. An informational approach to the global optimization of expensive-to-evaluate functions. *Journal of Global Optimization*, 2006.

[12] N. Srinivas, A. Krause, S. Kakade, and M. Seeger. Gaussian process optimization in the bandit setting: No regret and experimental design. In *ICML*, 2010.

[13] J. Mockus, V. Tiesis, and A. Zilinskas. The application of Bayesian methods for seeking the extremum. In L.C.W. Dixon and G.P. Szego, editors, *Towards Global Optimization*, volume 2, pages 117–129. North Holland, New York, 1978.

[14] C.E. Rasmussen and C. Williams. *Gaussian Processes for Machine Learning*.

[15] D. Ginsbourger, D. Dupuy, A. Badea, L. Carraro, and O. Roustant. A note on the choice and the estimation of kriging models for the analysis of deterministic computer experiments. 25:115–131, 2009.

[16] R. Bardenet and B. Kégl. Surrogating the surrogate: accelerating Gaussian Process optimization with mixtures. In *ICML*, 2010.

[17] P. Larrañaga and J. Lozano, editors. *Estimation of Distribution Algorithms: A New Tool for Evolutionary Computation*. Springer, 2001.

[18] N. Hansen. The CMA evolution strategy: a comparing review. In J.A. Lozano, P. Larranaga, I. Inza, and E. Bengoetxea, editors, *Towards a new evolutionary computation. Advances on estimation of distribution algorithms*, pages 75–102. Springer, 2006.

[19] J. Bergstra and Y. Bengio. Random search for hyper-parameter optimization. The Learning Workshop (Snowbird), 2011.

[20] A. Hyvärinen and E. Oja. Independent component analysis: Algorithms and applications. *Neural Networks*, 13(4–5):411–430, 2000.

[21] J. Bergstra and Y. Bengio. Random search for hyper-parameter optimization. JMLR, 2012. Accepted.

[22] C. Bishop. Neural networks for pattern recognition. 1995.

[23] J. Bergstra, O. Breuleux, F. Bastien, P. Lamblin, R. Pascanu, G. Desjardins, J. Turian, and Y. Bengio. Theano: a CPU and GPU math expression compiler. In *Proceedings of the Python for Scientific Computing Conference (SciPy)*, June 2010.

